# Non-Local Manifold Parzen Windows

**Yoshua Bengio, Hugo Larochelle and Pascal Vincent**
Dept. IRO, Université de Montréal
P.O. Box 6128, Downtown Branch, Montreal, H3C 3J7, Qc, Canada
{*bengioy,larocheh,vincentp*}*@iro.umontreal.ca*

## Abstract

To escape from the curse of dimensionality, we claim that one can learn non-local functions, in the sense that the value and shape of the learned function at $x$ must be inferred using examples that may be far from $x$. With this objective, we present a non-local non-parametric density estimator. It builds upon previously proposed Gaussian mixture models with regularized covariance matrices to take into account the local shape of the manifold. It also builds upon recent work on non-local estimators of the tangent plane of a manifold, which are able to generalize in places with little training data, unlike traditional, local, non-parametric models.

## 1  Introduction

A central objective of statistical machine learning is to discover structure in the joint distribution between random variables, so as to be able to make predictions about new combinations of values of these variables. A central issue in obtaining generalization is how information from the training examples can be used to make predictions about new examples and, without strong prior assumptions (i.e. in non-parametric models), this may be fundamentally difficult, as illustrated by the curse of dimensionality.

(Bengio, Delalleau and Le Roux, 2005) and (Bengio and Monperrus, 2005) present several arguments illustrating some fundamental limitations of modern kernel methods due to the curse of dimensionality, when the kernel is local (like the Gaussian kernel). These arguments are all based on the locality of the estimators, i.e., that very important information about the predicted function at $x$ is derived mostly from the near neighbors of $x$ in the training set. This analysis has been applied to supervised learning algorithms such as SVMs as well as to unsupervised manifold learning algorithms and graph-based semi-supervised learning. The analysis in (Bengio, Delalleau and Le Roux, 2005) highlights intrinsic limitations of such local learning algorithms, that can make them fail when applied on problems where one has to look beyond what happens locally in order to overcome the curse of dimensionality, or more precisely when the function to be learned has many variations while there exist more compact representations of these variations than a simple enumeration.

This strongly suggests to investigate **non-local learning methods**, which can in principle generalize at $x$ using information gathered at training points $x_i$ that are far from $x$. We present here such a non-local learning algorithm, in the realm of density estimation.

The proposed non-local non-parametric density estimator builds upon the Manifold Parzen density estimator (Vincent and Bengio, 2003) that associates a regularized Gaussian with

each training point, and upon recent work on non-local estimators of the tangent plane of a manifold (Bengio and Monperrus, 2005). The *local* covariance matrix characterizing the density in the immediate neighborhood of a data point is learned as a **function** of that data point, with *global* parameters. This allows to potentially generalize in places with little or no training data, unlike traditional, local, non-parametric models. Here, the implicit assumption is that there is some kind of regularity in the shape of the density, such that learning about its shape in one region could be informative of the shape in another region that is not adjacent. Note that the smoothness assumption typically underlying non-parametric models relies on a simple form of such transfer, but only for neighboring regions, which is not very helpful when the intrinsic dimension of the data (the dimension of the manifold on which or near which it lives) is high or when the underlying density function has many variations (Bengio, Delalleau and Le Roux, 2005). The proposed model is also related to the Neighborhood Component Analysis algorithm (Goldberger et al., 2005), which learns a global covariance matrix for use in the Mahalanobis distance within a non-parametric classifier. Here we generalize this global matrix to one that is a function of the datum $x$.

## 2  Manifold Parzen Windows

In the Parzen Windows estimator, one puts a spherical (*isotropic*) Gaussian around each training point $x_i$, with a single shared variance hyper-parameter. One approach to improve on this estimator, introduced in (Vincent and Bengio, 2003), is to use not just the presence of $x_i$ and its neighbors but also their geometry, trying to infer the principal characteristics of the local shape of the manifold (where the density concentrates), which can be summarized in the covariance matrix of the Gaussian, as illustrated in Figure 1. If the data concentrates in certain directions around $x_i$, we want that covariance matrix to be "flat" (near zero variance) in the orthogonal directions.

One way to achieve this is to parametrize each of these covariance matrices in terms of "principal directions" (which correspond to the tangent vectors of the manifold, if the data concentrates on a manifold). In this way we do not need to specify individually all the entries of the covariance matrix. The only required assumption is that the "noise directions" orthogonal to the "principal directions" all have the same variance.

$$\hat{p}(y) = \frac{1}{n} \sum_{i=1}^{n} N(y; x_i + \mu(x_i), S(x_i)) \tag{1}$$

where $N(y; x_i + \mu(x_i), S(x_i))$ is a Gaussian density at $y$, with mean vector $x_i + \mu(x_i)$ and covariance matrix $S(x_i)$ represented compactly by

$$S(x_i) = \sigma_{noise}^2(x_i)I + \sum_{j=1}^{d} s_j^2(x_i)v_j(x_i)v_j(x_i)' \tag{2}$$

where $s_j^2(x_i)$ and $\sigma_{noise}^2(x_i)$ are scalars, and $v_j(x_i)$ denotes a "principal" direction with variance $s_j^2(x_i) + \sigma_{noise}^2(x_i)$, while $\sigma_{noise}^2(x_i)$ is the noise variance (the variance in all the other directions). $v_j(x_i)'$ denotes the transpose of $v_j(x_i)$.

In (Vincent and Bengio, 2003), $\mu(x_i) = 0$, and $\sigma_{noise}^2(x_i) = \sigma_0^2$ is a global hyper-parameter, while $(\lambda_j(x_i), v_j) = (s_j^2(x_i) + \sigma_{noise}^2(x_i), v_j(x_i))$ are the leading (eigen-value,eigenvector) pairs from the eigen-decomposition of a locally weighted covariance matrix (e.g. the empirical covariance of the vectors $x_l - x_i$, with $x_l$ a near neighbor of $x_i$). The "noise level" hyper-parameter $\sigma_0^2$ must be chosen such that the principal eigenvalues are all greater than $\sigma_0^2$. Another hyper-parameter is the number $d$ of principal components to keep. Alternatively, one can choose $\sigma_{noise}^2(x_i)$ to be the $(d+1)^{th}$ eigenvalue, which guarantees that $\lambda_j(x_i) > \sigma_{noise}^2(x_i)$, and gets rid of a hyper-parameter. This very simple model was found to be consistently better than the ordinary Parzen density estimator in numerical experiments in which all hyper-parameters are chosen by cross-validation.

# 3 Non-Local Manifold Tangent Learning

In (Bengio and Monperrus, 2005) a manifold learning algorithm was introduced in which the tangent plane of a $d$-dimensional manifold at $x$ is learned as a function of $x \in \mathbb{R}^D$, using globally estimated parameters. The output of the predictor function $F(x)$ is a $d \times D$ matrix whose $d$ rows are the $d$ (possibly non-orthogonal) vectors that span the tangent plane. The training information about the tangent plane is obtained by considering pairs of near neighbors $x_i$ and $x_j$ in the training set. Consider the predicted tangent plane of the manifold at $x_i$, characterized by the rows of $F(x_i)$. For a good predictor we expect the vector $(x_i - x_j)$ to be close to its projection on the tangent plane, with local coordinates $w \in \mathbb{R}^d$. $w$ can be obtained analytically by solving a linear system of dimension $d$.

The training criterion chosen in (Bengio and Monperrus, 2005) then minimizes the sum over such $(x_i, x_j)$ of the sinus of the projection angle, i.e. $||F'(x_i)w - (x_j - x_i)||^2/||x_j - x_i||^2$. It is a heuristic criterion, which will be replaced in our new algorithm by one derived from the maximum likelihood criterion, considering that $F(x_i)$ indirectly provides the principal eigenvectors of the local covariance matrix at $x_i$. Both criteria gave similar results experimentally, but the model proposed here yields a complete density estimator. In both cases $F(x_i)$ can be interpreted as specifying the directions in which one expects to see the most variations when going from $x_i$ to one of its near neighbors in a finite sample.

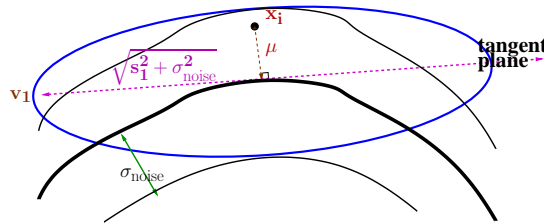

Figure 1: *Illustration of the local parametrization of local or Non-Local Manifold Parzen. The examples around training point $x_i$ are modeled by a Gaussian. $\mu(x_i)$ specifies the center of that Gaussian, which should be non-zero when $x_i$ is off the manifold. $v_k$'s are principal directions of the Gaussian and are tangent vectors of the manifold. $\sigma_{\text{noise}}$ represents the thickness of the manifold.*

# 4 Proposed Algorithm: Non-Local Manifold Parzen Windows

In equations (1) and (2) we wrote $\mu(x_i)$ and $S(x_i)$ as if they were **functions** of $x_i$ rather than simply using indices $\mu_i$ and $S_i$. This is because we introduce here a non-local version of Manifold Parzen Windows inspired from the non-local manifold tangent learning algorithm, i.e., in which we can **share information about the density across different regions of space**. In our experiments we use a neural network of $nhid$ hidden neurons, with $x_i$ in input to predict $\mu(x_i)$, $\sigma^2_{noise}(x_i)$, and the $s^2_j(x_i)$ and $v_j(x_i)$. The vectors computed by the neural network do not need to be orthonormal: we only need to consider the subspace that they span. Also, the vectors' squared norm is used to infer $s^2_j(x_i)$, instead of having a separate output for them. We will note $F(x_i)$ the matrix whose rows are the vectors output of the neural network. From it we obtain the $s^2_j(x_i)$ and $v_j(x_i)$ by performing a singular value decomposition, i.e. $F'F = \sum_{j=1}^{d} s^2_j v_j v'_j$. Moreover, to make sure $\sigma^2_{noise}$ does not get too small, which could make the optimization unstable, we impose $\sigma^2_{noise}(x_i) = s^2_{noise}(x_i) + \sigma^2_0$, where $s_{noise}(\cdot)$ is an output of the neural network and $\sigma^2_0$ is a fixed constant.

Imagine that the data were lying near a lower dimensional manifold. Consider a training example $x_i$ near the manifold. The Gaussian centered near $x_i$ tells us how neighbors of

$x_i$ are expected to differ from $x_i$. Its "principal" vectors $v_j(x_i)$ span the tangent of the manifold near $x_i$. The Gaussian center variation $\mu(x_i)$ tells us how $x_i$ is located with respect to its projection on the manifold. The noise variance $\sigma^2_{noise}(x_i)$ tells us how far from the manifold to expect neighbors, and the directional variances $s_j^2(x_i) + \sigma^2_{noise}(x_i)$ tell us how far to expect neighbors on the different local axes of the manifold, near $x_i$'s projection on the manifold. Figure 1 illustrates this in 2 dimensions.

The important element of this model is that the parameters of the predictive neural network can potentially represent non-local structure in the density, i.e., they allow to potentially discover shared structure among the different covariance matrices in the mixture. Here is the pseudo code algorithm for training Non-Local Manifold Parzen (NLMP):

---

**Algorithm NLMP::Train($X, d, k, k_\mu, \mu(\cdot), S(\cdot), \sigma_0^2$)**

**Input**: training set $X$, chosen number of principal directions $d$, chosen number of neighbors $k$ and $k_\mu$, initial functions $\mu(\cdot)$ and $S(\cdot)$, and regularization hyper-parameter $\sigma_0^2$.

**(1)** For $x_i \in X$

**(2)** Collect $\max(k, k_\mu)$ nearest neighbors of $x_j$.
Below, call $y_j$ one of the $k$ nearest neighbors, $y_j^\mu$ one of the $k_\mu$ nearest neighbors.

**(3)** Perform a stochastic gradient step on parameters of $S(\cdot)$ and $\mu(\cdot)$,
using the negative log-likelihood error signal on the $y_j$, with a Gaussian
of mean $x_i + \mu(x_i)$ and of covariance matrix $S(x_i)$.

The approximate gradients are:

$$\frac{\partial C(y_j^\mu, x_i)}{\partial \mu(x_i)} = -\frac{1}{n_{k_\mu}(y_j^\mu)} S(x_i)^{-1}(y_j^\mu - x_i - \mu(x_i))$$

$$\frac{\partial C(y_j, x_i)}{\partial \sigma^2_{noise}(x_i)} = 0.5 \frac{1}{n_k(y_j)} \left( Tr(S(x_i)^{-1}) - ||(y_j - x_i - \mu(x_i))'S(x_i)^{-1}||^2 \right)$$

$$\frac{\partial C(y_j, x_i)}{\partial F(x_i)} = \frac{1}{n_k(y_j)} F(x_i) S(x_i)^{-1} \left( I - (y_j - x_i - \mu(x_i))(y_j - x_i - \mu(x_i))'S(x_i)^{-1} \right)$$

where $n_k(y) = |\mathcal{N}_k(y)|$ is the number of points in the training set that
have $y$ among their $k$ nearest neighbors.

**(4)** Go to **(1)** until a given criterion is satisfied (e.g. average NLL of NLMP density estimation on a validation set stops decreasing)

**Result**: trained $\mu(\cdot)$ and $S(\cdot)$ functions, with corresponding $\sigma_0^2$.

---

Deriving the gradient formula (the derivative of the log-likelihood with respect to the neural network outputs) is lengthy but straightforward. The main trick is to do a Singular Value Decomposition of the basis vectors computed by the neural network, and to use known simplifying formulas for the derivative of the inverse of a matrix and of the determinant of a matrix. Details on the gradient derivation and on the optimization of the neural network are given in the technical report (Bengio and Larochelle, 2005).

## 5 Computationally Efficient Extension: Test-Centric NLMP

While the NLMP algorithm appears to perform very well, one of its main practical limitation for density estimation, that it shares with Manifold Parzen, is the large amount of computation required upon testing: for *each* test point $x$, the complexity of the computation is $O(n.d.D)$ (where $D$ is the dimensionality of input space $\mathbb{R}^D$).

However there may be a different and cheaper way to compute an estimate of the density at $x$. We build here on an idea suggested in (Vincent, 2003), which yields an estimator that

does not exactly integrate to one, but this is not an issue if the estimator is to be used for applications such as classification. Note that in our presentation of NLMP, we are using "hard" neighborhoods (i.e. a local weighting kernel that assigns a weight of 1 to the $k$ nearest neighbors and 0 to the rest) but it could easily be generalized to "soft" weighting, as in (Vincent, 2003).

Let us decompose the true density at $x$ as: $p(x) = p(x|x \in B_k(x))P(B_k(x))$, where $B_k(x)$ represents the spherical ball centered on $x$ and containing the $k$ nearest neighbors of $x$ (i.e., the ball with radius $\|x - N_k(x)\|$ where $N_k(x)$ is the $k$-th neighbor of $x$ in the training set).

It can be shown that the above NLMP learning procedure looks for functions $\mu(\cdot)$ and $S(\cdot)$ that best characterize the distribution of the $k$ training-set nearest neighbors of $x$ as the normal $N(\cdot; x + \mu(x), S(x))$. If we trust this locally normal (unimodal) approximation of the neighborhood distribution to be appropriate then we can approximate $p(x|x \in B_k(x))$ by $N(x; x + \mu(x), S(x))$. The approximation should be good when $B_k(x)$ is small and $p(x)$ is continuous. Moreover as $B_k(x)$ contains $k$ points among $n$ we can approximate $P(B_k(x))$ by $\frac{k}{n}$.

This yields the estimator $\hat{p}(x) = N(x; x + \mu(x), S(x))\frac{k}{n}$, which requires only $O(d.D)$ time to evaluate at a test point. We call this estimator *Test-centric NLMP*, since it considers only the Gaussian predicted at the test point, rather than a mixture of all the Gaussians obtained at the training points.

## 6   Experimental Results

We have performed comparative experiments on both toy and real-world data, on density estimation and classification tasks. All hyper-parameters are selected by cross-validation, and the costs on a large test set is used to compare final performance of all algorithms.

**Experiments on toy 2D data**. To understand and validate the non-local algorithm we tested it on toy 2D data where it is easy to understand what is being learned. The **sinus** data set includes examples sampled around a sinus curve. In the **spiral** data set examples are sampled near a spiral. Respectively, 57 and 113 examples are used for training, 23 and 48 for validation (hyper-parameter selection), and 920 and 3839 for testing. The following algorithms were compared:
• Non-Local Manifold Parzen Windows. The hyper-parameters are the number of principal directions (i.e., the dimension of the manifold), the number of nearest neighbors $k$ and $k_\mu$, the minimum constant noise variance $\sigma_0^2$ and the number of hidden units of the neural network.
• Gaussian mixture with full but regularized covariance matrices. Regularization is done by setting a minimum constant value $\sigma_0^2$ to the eigenvalues of the Gaussians. It is trained by EM and initialized using the k-means algorithm. The hyper-parameter is $\sigma_0^2$, and early stopping of EM iterations is done with the validation set.
• Parzen Windows density estimator, with a spherical Gaussian kernel. The hyper-parameter is the spread of the Gaussian kernel.
• Manifold Parzen density estimator. The hyper-parameters are the number of principal components, $k$ of the nearest neighbor kernel and the minimum eigenvalue $\sigma_0^2$.

Note that, for these experiments, the number of principal directions (or components) was fixed to 1 for both NLMP and Manifold Parzen.

Density estimation results are shown in table 1. To help understand why Non-Local Manifold Parzen works well on these data, figure 2 illustrates the learned densities for the sinus and spiral data. Basically, it works better here because it yields an estimator that is less sensitive to the specific samples around each test point, thanks to its ability to share structure

| Algorithm | sinus | spiral |
|---|---|---|
| **Non-Local MP** | **1.144** | **-1.346** |
| Manifold Parzen | 1.345 | -0.914 |
| Gauss Mix Full | 1.567 | -0.857 |
| Parzen Windows | 1.841 | -0.487 |

Table 1: *Average out-of-sample negative log-likelihood on two toy problems, for Non-Local Manifold Parzen, a Gaussian mixture with full covariance, Manifold Parzen, and Parzen Windows. The non-local algorithm dominates all the others.*

| Algorithm | Valid. | Test |
|---|---|---|
| **Non-Local MP** | **-73.10** | **-76.03** |
| Manifold Parzen | 65.21 | 58.33 |
| Parzen Windows | 77.87 | 65.94 |

Table 2: *Average Negative Log-Likelihood on the digit rotation experiment, when testing on a digit class (1's) not used during training, for Non-Local Manifold Parzen, Manifold Parzen, and Parzen Windows. The non-local algorithm is clearly superior.*

across the whole training set.

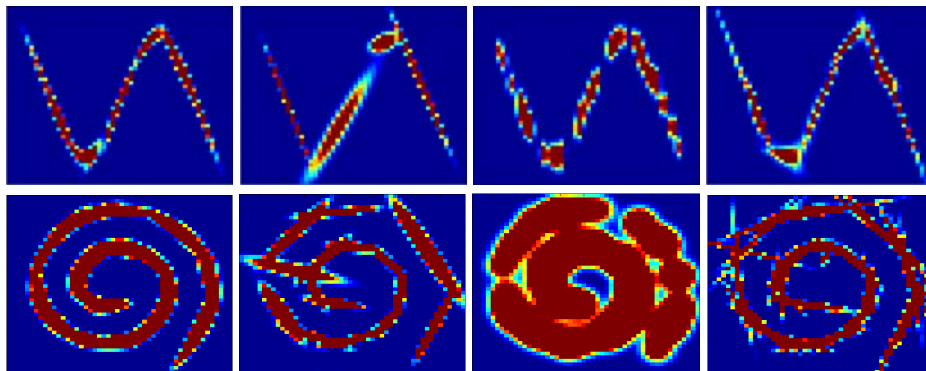

Figure 2: *Illustration of the learned densities (sinus on top, spiral on bottom) for four compared models. From left to right: Non-Local Manifold Parzen, Gaussian mixture, Parzen Windows, Manifold Parzen. Parzen Windows wastes probability mass in the spheres around each point, while leaving many holes. Gaussian mixtures tend to choose too few components to avoid overfitting. The Non-Local Manifold Parzen exploits global structure to yield the best estimator.*

**Experiments on rotated digits**. The next experiment is meant to show both qualitatively and quantitatively the power of non-local learning, by using 9 classes of rotated digit images (from 729 first examples of the USPS training set) to learn about the rotation manifold and testing on the left-out class (digit 1), not used for training. Each training digit was rotated by 0.1 and 0.2 radians and all these images were used as training data. We used NLMP for training, and for testing we formed an augmented mixture with Gaussians centered not only on the training examples, but also on the original unrotated 1 digits. We tested our estimator on the rotated versions of each of the 1 digits. We compared this to Manifold Parzen trained on the training data containing both the original and rotated images of the training class digits and the unrotated 1 digits. The objective of the experiment was to see if the model was able to infer the density correctly around the original unrotated images, i.e., to predict a high probability for the rotated versions of these images. In table 2 we see quantitatively that the non-local estimator predicts the rotated images much better.

As qualitative evidence, we used small steps in the principal direction predicted by *Test-centric NLMP* to rotate an image of the digit 1. To make this task even more illustrative of the generalization potential of non-local learning, we followed the tangent in the direction opposite to the rotations of the training set. It can be seen in figure 3 that the rotated

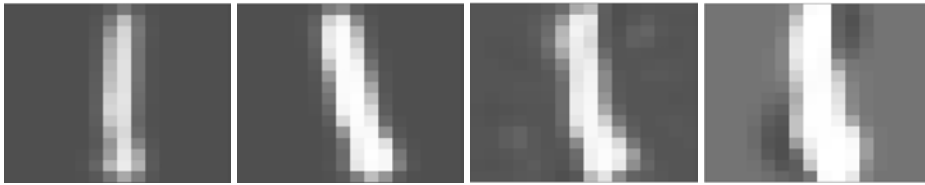

Figure 3: *From left to right: original image of a digit 1; rotated analytically by* $-0.2$ *radians; Rotation predicted using Non-Local MP; rotation predicted using MP. Rotations are obtained by following the tangent vector in small steps.*

digit obtained is quite similar to the same digit analytically rotated. For comparison, we tried to apply the same rotation technique to that digit, but by using the principal direction, computed by Manifold Parzen, of its nearest neighbor's Gaussian component in the training set. This clearly did not work, and hence shows how crucial non-local learning is for this task.

In this experiment, to make sure that NLMP focusses on the tangent plane of the rotation manifold, we fixed the number of principal directions $d = 1$ and the number of nearest neighbors $k = 1$, and also imposed $\mu(\cdot) = 0$. The same was done for Manifold Parzen.

**Experiments on Classification by Density Estimation.** The USPS data set was used to perform a classification experiment. The original training set (7291) was split into a training (first 6291) and validation set (last 1000), used to tune hyper-parameters. One density estimator for each of the 10 digit classes is estimated. For comparison we also show the results obtained with a Gaussian kernel Support Vector Machine (already used in (Vincent and Bengio, 2003)). **Non-local MP\*** refers to the variation described in (Bengio and Larochelle, 2005), which attemps to train faster the components with larger variance. The t-test statistic for the null hypothesis of no difference in the average classification error on the test set of 2007 examples between Non-local MP and the strongest competitor (Manifold Parzen) is shown in parenthesis. Figure 4 also shows some of the invariant transformations learned by **Non-local MP** for this task.

Note that better SVM results (about 3% error) can be obtained using prior knowledge about image invariances, e.g. with virtual support vectors (Decoste and Scholkopf, 2002). However, as far as we know the NLMP performance is the best on the original USPS dataset among algorithms that do not use prior knowledge about images.

| Algorithm | Valid. | Test | Hyper-Parameters |
|---|---|---|---|
| SVM | 1.2% | 4.68% | $C = 100, \sigma = 8$ |
| Parzen Windows | 1.8% | 5.08% | $\sigma = 0.8$ |
| Manifold Parzen | 0.9% | 4.08% | $d = 11, k = 11, \sigma_0^2 = 0.1$ |
| **Non-local MP** | **0.6%** | **3.64% (-1.5218)** | $d = 7, k = 10, k_\mu = 10,$ $\sigma_0^2 = 0.05, n_{hid} = 70$ |
| **Non-local MP\*** | **0.6%** | **3.54% (-1.9771)** | $d = 7, k = 10, k_\mu = 4,$ $\sigma_0^2 = 0.05, n_{hid} = 30$ |

Table 3: *Classification error obtained on USPS with SVM, Parzen Windows and Local and Non-Local Manifold Parzen Windows classifiers. The hyper-parameters shown are those selected with the validation set.*

## 7    Conclusion

We have proposed a non-parametric density estimator that, unlike its predecessors, is able to generalize far from the training examples by capturing global structural features of the

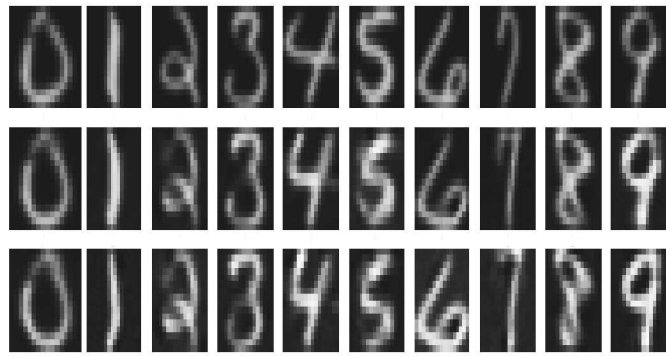

Figure 4: *Tranformations learned by* **Non-local MP**. *The top row shows digits taken from the USPS training set, and the two following rows display the results of steps taken by one of the 7 principal directions learned by* **Non-local MP**, *the third one corresponding to more steps than the second one.*

density. It does so by learning a function with global parameters that successfully predicts the local shape of the density, i.e., the tangent plane of the manifold along which the density concentrates. Three types of experiments showed that this idea works, yields improved density estimation and reduced classification error compared to its local predecessors.

**Acknowledgments**

The authors would like to thank the following funding organizations for support: NSERC, MITACS, and the Canada Research Chairs. The authors are also grateful for the feedback and stimulating exchanges that helped to shape this paper, with Sam Roweis and Olivier Delalleau.

# References

Bengio, Y., Delalleau, O., and Le Roux, N. (2005). The curse of dimensionality for local kernel machines. Technical Report 1258, Département d'informatique et recherche opérationnelle, Université de Montréal.

Bengio, Y. and Larochelle, H. (2005). Non-local manifold parzen windows. Technical report, Département d'informatique et recherche opérationnelle, Université de Montréal.

Bengio, Y. and Monperrus, M. (2005). Non-local manifold tangent learning. In Saul, L., Weiss, Y., and Bottou, L., editors, *Advances in Neural Information Processing Systems 17*. MIT Press.

Decoste, D. and Scholkopf, B. (2002). Training invariant support vector machines. *Machine Learning*, 46:161–190.

Goldberger, J., Roweis, S., Hinton, G., and Salakhutdinov, R. (2005). Neighbourhood component analysis. In Saul, L., Weiss, Y., and Bottou, L., editors, *Advances in Neural Information Processing Systems 17*. MIT Press.

Vincent, P. (2003). *Modèles à Noyaux à Structure Locale*. PhD thesis, Université de Montréal, Département d'informatique et recherche opérationnelle, Montreal, Qc., Canada.

Vincent, P. and Bengio, Y. (2003). Manifold parzen windows. In Becker, S., Thrun, S., and Obermayer, K., editors, *Advances in Neural Information Processing Systems 15*, Cambridge, MA. MIT Press.
